# Active Noise Canceling using Analog Neuro-Chip with On-Chip Learning Capability

**Jung-Wook Cho and Soo-Young Lee**
Computation and Neural Systems Laboratory
Department of Electrical Engineering
Korea Advanced Institute of Science and Technology
373-1 Kusong-dong, Yusong-gu, Taejon 305-701, Korea
sylee@ee.kaist.ac.kr

## Abstract

A modular analogue neuro-chip set with on-chip learning capability is developed for active noise canceling. The analogue neuro-chip set incorporates the error backpropagation learning rule for practical applications, and allows pin-to-pin interconnections for multi-chip boards. The developed neuro-board demonstrated active noise canceling without any digital signal processor. Multi-path fading of acoustic channels, random noise, and nonlinear distortion of the loud speaker are compensated by the adaptive learning circuits of the neuro-chips. Experimental results are reported for cancellation of car noise in real time.

## 1 INTRODUCTION

Both analog and digital implementations of neural networks have been reported. Digital neuro-chips can be designed and fabricated with the help of well-established CAD tools and digital VLSI fabrication technology [1]. Although analogue neuro-chips have potential advantages on integration density and speed over digital chips[2], they suffer from non-ideal characteristics of the fabricated chips such as offset and nonlinearity, and the fabricated chips are not flexible enough to be used for many different applications. Also, much careful design is required, and the fabricated chip characteristics are fairly dependent upon fabrication processes.

For the implementation of analog neuro-chips, there exist two different approaches, i.e., with and without on-chip learning capability [3,4]. Currently the majority of analog neuro-chips does not have learning capability, while many practical applications require on-line adaptation to continuously changing environments, and must have on-line adaptation learning capability. Therefore neuro-chips with on-chip learning capability are essential for such practical applications. Modular architecture is also

advantageous to provide flexibility of implementing many large complex systems from same chips.

Although many applications have been studied for analog neuro-chips, it is very important to find proper problems where analog neuro-chips may have potential advantages over popular DSPs. We believe applications with analog input/output signals and high computational requirements are those good problems. For example, active noise controls [5] and adaptive equalizers [6,7] are good applications for analog neuro-chips.

In this paper we report a demonstration of the active noise canceling, which may have many applications in real world. A modular analog neuro-chip set is developed with on-chip learning capability, and a neuro-board is fabricated from multiple chips with PC interfaces for input and output measurements. Unlike our previous implementations for adaptive equalizers with binary outputs [7], both input and output values are analogue in this noise canceling.

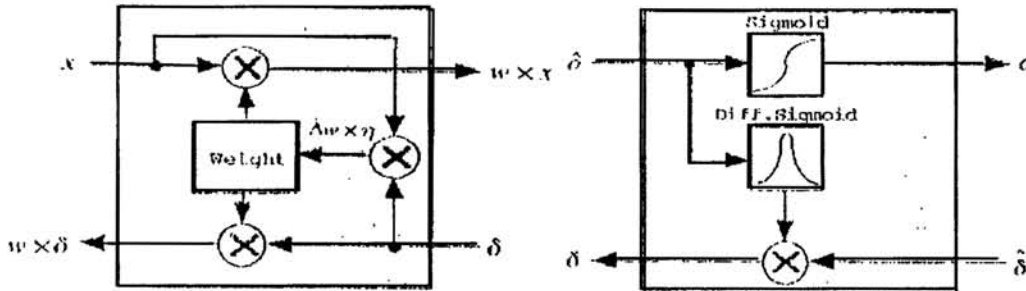

**Figure 1. Block diagram of a synapse cell**   **Figure 2. Block diagram of a neuron cell**

## 2   ANALOG NEURO-CHIP WITH ON-CHIP LEARNING

We had developed analog neuro-chips with error backpropagation learning capability. With the modular architecture the developed analog neuro-chip set consists of a synapse chip and a neuron chip.[8] The basic cell of the synapse chip is shown in Figure 1. Each synapse cell receives two inputs, i.e., pre-synaptic neural activation $x$ and error correction term $\delta$, and generates two outputs, i.e., feed-forward signal $wx$ and back-propagated error $w\delta$. Also it updates a stored weight $w$ by the amount of $x\delta$. Therefore, a synapse cell consists of three multiplier circuits and one analogue storage for the synaptic weight. Figure 2 shows the basic cell in the neuron chip, which collects signals from synapses in the previous layer and distributes to synapses in the following layer. Each neuron body receives two inputs, i.e., post-synaptic neural activation $\hat{o}$ and back-propagated error $\delta$ from the following layer, and generates two outputs, i.e., Sigmoid-squashed neural activation $o$ and a new backpropagated error $\delta$ multiplied by a bell-shaped Sigmoid-derivative. The backpropagated error may be input to the synapse cells in the previous layer.

To provide easy connectivity with other chips, the two inputs of the synapse cell are represented as voltage, while the two outputs are as currents for simple current summation. On the other hand the inputs and outputs of the neuron cell are represented as currents and voltages, respectively. For simple pin-to-pin connections between chips, one package pin is maintained to each input and output of the chip. No time-

multiplexing is introduced, and no other control is required for multi-chip and multi-layer systems. However, it makes the number of package pins the main limiting factor for the number of synapse and neuron cells in the developed chip sets.

Although many simplified multipliers had been reported for high-density integration, their performance is limited in linearity, resolution, and speed. For on-chip learning, it is desirable to have high precision, and a faithful implementation of the 4-quadranr Gilbert multipliers is used. Especially, the multiplier for weight updates in the synapse cell requires high precision.[9] The synaptic weight is stored on a capacitor, and an MOS switch is used to allow current flow from the multiplier to the capacitor during a short time interval for weight adaptation. For applications like active noise controls [5] and telecommunications [6,7], tapped analog delay lines are also designed and integrated in the synapse chip. To reduce offset accumulation, a parallel analog delay line is adopted. Same offset voltage is introduced for operational amplifiers at all nodes [10]. Diffusion capacitors with 2.2 pF are used for the storage of the tapped analog delay line.

In a synapse chip 250 synapse cells are integrated in a 25x10 array with a 25-tap analog delay line. Inputs may be applied either from the analog delay line or from external pins in parallel. To select a capacitor in the cell for refresh, decoders are placed in columns and rows. The actual size of the synapse cell is 141μm x 179μm, and the size of the synapse chip is 5.05mm x 5.05mm. The chip is fabricated in a 0.8μm single-poly CMOS process. On the other hand, the neuron chip has a very simple structure, which consists of 20 neuron cells without additional circuits. The Sigmoid circuit [3] in the neuron cell uses a differential pair, and the slope and amplitude are controlled by a voltage-controlled resistor [11]. Sigmoid-derivative circuit is also using differential pair with min-select circuit. The size of the neuron cell is 177.2μm x 62.4μm.

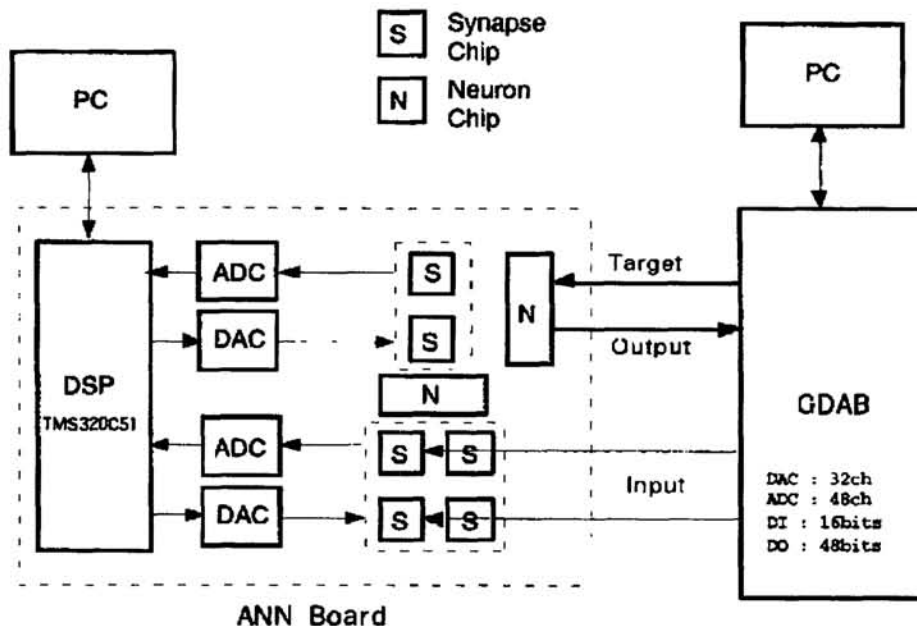

**Figure 3: Block diagram of the analog neuro-board**

Using these chip sets, an analog neuro-system is constructed. Figure 3 shows a brief block diagram of the analog neuro-system, where an analogue neuro-board is interfaced to a host computer through a GDAB (General Data Acquisition Board). The GDAB board is specially designed for the data interface with the analogue neuro-chips. The neuro-board has 6 synapse chips and 2 neuron chips with the 2-layer Perceptron architecture. For test and development purposes, a DSP, ADC and DAC are installed on the neuro-board to refresh and adjust weights.

Forward propagation time of the 2 layers Perceptron is measured as about 30 μsec. Therefore the computation speed of the neuro-board is about 266 MCPS (Mega Connections Per Second) for recall and about 200 MCUPS (Mega Connections Updates Per Second) for error backpropagation learning. To achieve this speed with a DSP, about 400 MIPS is required for recall and at least 600 MIPS for error-back propagation learning.

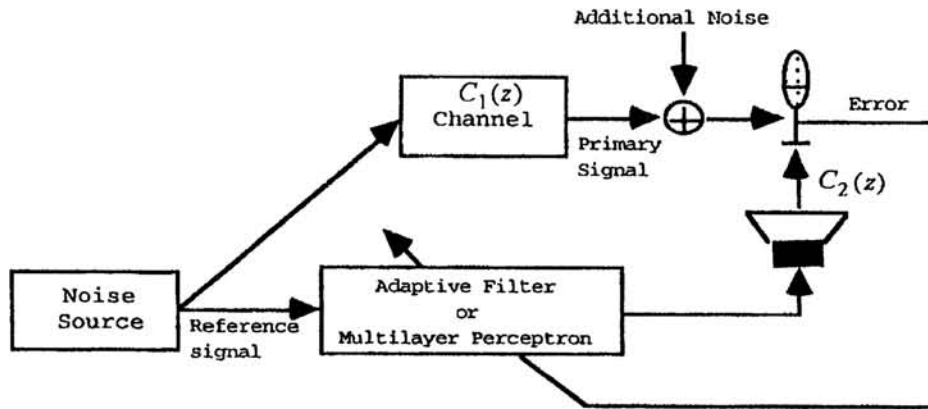

**Figure 4: Structure of a feedforward active noise canceling**

## 3 ACTIVE NOISE CANCELING USING NEURO-CHIP

Basic architecture of the feedforward active noise canceling is shown in Figure 4. An area near the microphone is called "quiet zone," which actually means noise should be small in this area. Noise propagates from a source to the quiet zone through a dispersive medium, of which characteristics are modeled as a finite impulse response (FIR) filter with additional random noise. An active noise canceller should generate electric signals for a loud speaker, which creates acoustic signals to cancel the noise at the quiet zone. In general the electric-to-acoustic signal transfer characteristics of the loud speaker is nonlinear, and the overall active noise canceling (ANC) system also becomes nonlinear. Therefore, multilayer Perceptron has a potential advantage over popular transversal adaptive filters based on linear-mean.-square (LMS) error minimization.

Experiments had been conducted for car noise canceling. The reference signal for the noise source was extracted from an engine room, while a compact car was running at 60 km/hour. The difference of the two acoustic channels, i.e., $H(z) = C_1(z)/C_2(z)$, addition noise $n$, and nonlinear characteristics of the loud speaker need be compensated. Two different acoustic channels are used for the experiments. The first channel $H_1(z) = 0.894 + 0.447z^{-1}$ is a minimum phase channel, while the second non-

minimum phase channel $H_2(z) = 0.174 + 0.6z^{-1} + 0.6z^{-2} + 0.174z^{-3}$ characterizes frequency-selective multipath fading with a deep spectral amplitude null. A simple cubic distortion model was used for the characteristics of the loud speaker.[12] To compare performance of the neuro-chip with digital processors, computer simulation was first conducted with error backpropagation algorithm for a single hidden-layer Perceptron as well as the LMS algorithm for a transversal adaptive filter. Then, the same experimental data were provided to the developed neuro-board by a personal computer through the GDAB.

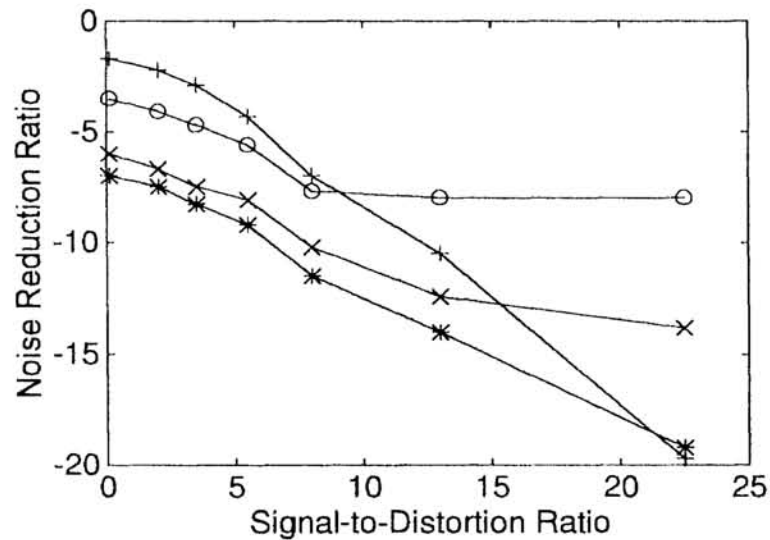

(a)

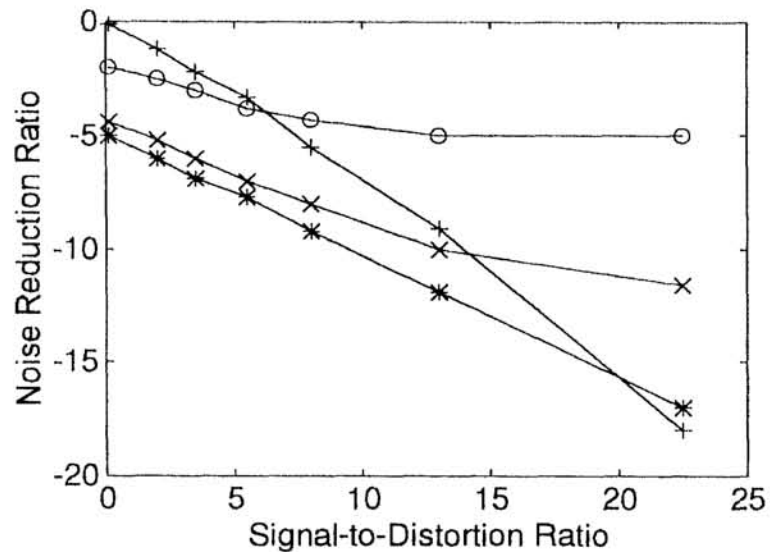

(b)

**Figure 5: Noise Reduction Ratio (dB) versus Signal-to-Distortion Ratio (dB) for (a) a simple acoustic channel $H_1(z)$ and (b) a multi-path fading acoustic channel $H_2(z)$.**

**Here, '+', '*', 'x', and 'o' denote results of LMS algorithm, neural networks simulation, neural network simulation with 8-bit input quantization, and neuro-chips, respectively.**

Results for the channels $H_1(z)$ and $H_2(z)$ are shown in Figures 5(a) and 5(b), respectively. Each point in these figures denotes the result of one experiment with different parameters. The horizontal axes represent Signal-to-Distortion Ratio (SDR) of the speaker nonlinear characteristics. The vertical axes represent Noise Reduction Ratio (NRR) of the active noise canceling systems. As expected, severe nonlinear distortion of the loud speaker resulted in poor noise canceling for the LMS canceller. However, the performance degradation was greatly reduced by neural network canceller. With the neuro-chips the performance was worse than that of computer simulation. Although the neuro-chip demonstrated active noise canceling and worked better than LMS cancellers for very small SDRs, i.e., very high nonlinear distortions, its performance became saturated to -8 dB and -5 dB NRRs, respectively. The performance saturation was more severe for the harder problem with the complicated $H_2(z)$ channel.

The performance degradation with neuro-chips may come from inherent limitations of analogue chips such as limited dynamic ranges of synaptic weights and signals, unwanted offsets and nonlinearity, and limited resolution of the learning rate and sigmoid slope.[9] However, other side effects of the GDAB board, i.e., fixed resolution of A/D converters and D/A converters for data I/O, also contributed to the performance degradation. The input and output resolutions of the GDAB were 16 bit and 8 bit, respectively. Unlike actual real-world systems the input values of the experimental analogue neuro-chips are these 8-bit quantized values. As shown in Figures 5, results of the computer simulation with 8-bit quantized target values showed much degraded performance compared to the floating-point simulations. Therefore, a significant portion of the poor performance in the experimental analogue system may be contributed from the A/D converters, and the analogue system may work better in real world systems.

Actual acoustic signals are plotted in Figure 6. The top, middle, and bottom signals denote noise , negated speaker signal, and residual noise at the quiet zone, respectively.

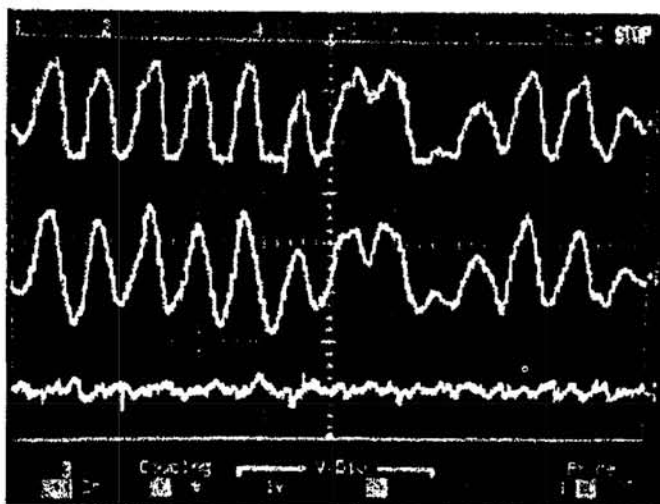

**Figure 6: Examples of noise, negated loud-speaker canceling signal, and residual error**

## 4  CONCLUSION

In this paper we report an experimental results of active noise canceling using analogue neuro-chips with on-chip learning capability. Although the its performance is limited due to nonideal characteristics of analogue chip itself and also peripheral devices, it clearly demonstrates feasibility of analogue chips for real world applications.

### Acknowledgements

This research was supported by Korean Ministry of Information and Tele-communications.

### References

[1] T. Watanabe, K. Kimura, M. Aoki, T. Sakata & K. Ito (1993) A Single 1.5-V Digital Chip for a 106 Synapse Neural Network, *IEEE Trans. Neural Network*, Vol.4, No.3, pp.387-393.

[2] T. Morie and Y. Amemiya (1994) An All-Analog Expandable Neural Network LSI with On-Chip Backpropagation Learning, *IEEE Journal of Solid State Circuits*, vol.29, No.9, pp.1086-1093.

[3] J.-W. Cho, Y. K. Choi, S.-Y. Lee (1996) Modular Neuro-Chip with On-Chip Learning and Adjustable Learning Parameters, *Neural Processing Letters*, Vol.4, No.1.

[4] J. Alspector, A. Jayakumar, S. Luna (1992) Experimental evaluation of learning in neural microsystem, *Advances in Neural Information Processing Systems* **4**, pp. 871-878.

[5] B. Widrow, *et al.* (1975) Adative Noise Cancelling: Principles and Applications, *Proceeding of IEEE*, Vol.63, No.12, pp.1692-1716.

[6] J. Choi, S.H. Bang, B.J. Sheu (1993) A Programmable Analog VLSI Neural Network Processor for Communication Receivers, *IEEE Transaction on Neural Network*, Vol.4, No.3, pp.484-495.

[7] J.-W. Cho and S.-Y. Lee (1998) Analog neuro-chips with on-chip learning capability for adaptive nonlinear equalizer, *Proc. IJCNN*, pp. 581-586, May 4-9, Anchorage, USA.

[8] J. Van der Spiegel, C. Donham, R. Etienne-Cummings, S. Fernando (1994) Large scale analog neural computer with programmable architecture and programmable time constants for temporal pattern analysis, *Proc. ICNN, pp. 1830-1835.*

[9] Y.K. Choi, K.H. Ahn, and S.Y. Lee (1996) Effects of multiplier offsets on on-chip learning for analog neuro-chip, *Neural Processing Letters*, vol. 4, No.1, 1-8.

[10] T. Enomoto, T. Ishihara and M. Yasumoto (1982) Integrated tapped MOS analogue delay line using switched-capacitor technique, *Electronics Lertters*, Vol.18, pp.193-194.

[11] P.B. Allen, D.R. Holberg (1987) *CMOS Analog Circuit Design*, Holt, Douglas Rinehart and Winston.

[12] F. Gao and W.M. Snelgrove (1991) Adaptive linearization of a loudspeaker, *Proc. International Conference on Acoustics, Speech and Signal processing*, pp. 3589-3592.